# Clustering via LP-based Stabilities

**Nikos Komodakis**
University of Crete
komod@csd.uoc.gr

**Nikos Paragios**
Ecole Centrale de Paris
INRIA Saclay Ile-de-France
nikos.paragios@ecp.fr

**Georgios Tziritas**
University of Crete
tziritas@csd.uoc.gr

## Abstract

A novel center-based clustering algorithm is proposed in this paper. We first formulate clustering as an NP-hard linear integer program and we then use linear programming and the duality theory to derive the solution of this optimization problem. This leads to an efficient and very general algorithm, which works in the dual domain, and can cluster data based on an arbitrary set of distances. Despite its generality, it is independent of initialization (unlike EM-like methods such as K-means), has guaranteed convergence, can automatically determine the number of clusters, and can also provide online optimality bounds about the quality of the estimated clustering solutions. To deal with the most critical issue in a center-based clustering algorithm (selection of cluster centers), we also introduce the notion of *stability* of a cluster center, which is a well defined LP-based quantity that plays a key role to our algorithm's success. Furthermore, we also introduce, what we call, the *margins* (another key ingredient in our algorithm), which can be roughly thought of as dual counterparts to stabilities and allow us to obtain computationally efficient approximations to the latter. Promising experimental results demonstrate the potentials of our method.

## 1 Introduction

Clustering is considered as one of the most fundamental unsupervised learning problems. It lies at the heart of many important tasks in machine learning, patter recognition, computer vision, data mining, biology, marketing, just to mention a few of its application areas. Most of the clustering methods are center-based, thus trying to extract a set of cluster centers that best 'describe' the input data. Typically, this translates into an optimization problem where one seeks to assign each input data point to a unique cluster center such that the total sum of the corresponding distances is minimized. These techniques are extremely popular and they are thus essential even to other types of clustering algorithms such as Spectral Clustering methods [1],[2].

Currently, most center-based clustering methods rely on EM-like schemes for optimizing their clustering objective function [3]. K-means is the most characteristic (and perhaps the most widely used) technique from this class. It keeps greedily refining a current set of cluster centers based on a simple gradient descent scheme. As a result, it can very easily get trapped to bad local minima and is extremely sensitive to initialization. It is thus likely to fail in problems with, e.g., a large number of clusters. A second very important drawback of many center-based clustering methods, which severely limits their applicability, is that they either require the input data to be of vectorial form and/or impose strong restrictions on the type of distance functions they can handle. Ideally, one would like to be able to cluster data based on arbitrary distances. This is an important point because, by an appropriate choice of these distances, clustering results with completely different characteristics can be achieved [4]. In addition to that, one would prefer that the number of clusters is automatically estimated by the algorithm (e.g., as a byproduct of the optimization process) and not given as input. In contrast to that, however, many algorithms assume that this number is known a priori.

To circumvent all the issues mentioned above, a novel center-based clustering algorithm is proposed in this paper. Similarly to other methods, it reduces clustering to a well-defined (but NP-hard) minimization problem, where, of course, the challenge now is how to obtain solutions of minimum objective value. To this end, we rely on the fact that the above problem admits a linear integer programming formulation. By making heavy use of a dual LP relaxation to that program, we then manage to derive a dual based algorithm for clustering. As in all center-based clustering techniques, the most critical component in the resulting algorithm is deciding what cluster centers to choose. To this end, we introduce, what we call, the stability of a data point as a cluster center (this is an LP-based quantity), which we consider as another contribution of this work. Intuitively, the stability of a data point as a cluster center tries to measure how much we need to penalize that point (by appropriately modifying the objective function) such that it can no longer be chosen as a center in an optimal solution of the modified problem. Obviously, one would like to choose as centers those points having high stability. For applying this idea in practice, however, a crucial issue that one needs to deal with is how to efficiently approximate these stability measures. To this end, we introduce, what we call, the margins, another very important concept in our algorithm and a key contribution of our work. As we prove in this paper, margins can be considered as dual to stabilities. Furthermore, they allow us to approximate the latter on the fly, i.e., as our algorithm runs. The outcome is an efficient and very easily implementable optimization algorithm, which works in the dual domain by iteratively updating a dual solution via two very simple operations: DISTRIBUTE and PROJECT. It can cluster data based on an arbitrary set of distances, which is the only input required by the algorithm (as a result, it can find use in a wide variety of applications, even in case where non-vectorial data need to be used). Furthermore, an important point is that, despite its generality, it does not get trapped to bad local minima. It is thus insensitive to initialization and can always compute clusterings of very low cost. Similarly to [5], the number of clusters does not need to be predefined, but is decided on the fly during the optimization process. However, unlike [5], convergence of the proposed method is always guaranteed and no parameters' adjustment needs to take place for this. Finally, an additional advantage of our method is that it can provide online optimality guarantees, which can be used for assessing the quality of the generated clusterings. These guarantees come in the form of lower bounds on the cost of the optimal clustering and are computed (for free) by simply using the cost of the dual solutions generated during the course of the algorithm.

## 2 Clustering via stabilities based on Linear Programming

Given a set of objects $\mathcal{V}$ with distances $\mathbf{d} = \{d_{pq}\}$, clustering amounts to choosing a set of cluster centers from $\mathcal{V}$ (say $\{q_i\}_{i=1}^k$) such that the sum of distances between each object and its closest center is minimized. To this end, we are going to use the following objective function $E(\cdot)$ (which will be referred to as the *primal cost* hereafter):

$$\min_{k,\{q_i\}_{i=1}^k} E(\{q_i\}_{i=1}^k) = \sum_{p \in \mathcal{V}} \min_i d_{pq_i} + \sum_i d_{q_i q_i} \tag{1}$$

Note that, in this case, we require that each cluster is chosen from the set $\mathcal{V}$. Also note that, besides $\{q_i\}$, here we optimize over the number of cluster centers $k$ as well. Of course, to avoid the trivial solution of choosing all objects as centers, we regularize the problem by assigning a penalty $d_{qq}$ to each chosen center $q$. Problem (1) has an equivalent formulation as a $0 - 1$ linear integer program [6], whose relaxation leads to the following LP (denoted by PRIMAL hereafter):

$$\text{PRIMAL} \equiv \min \sum_{p,q \in \mathcal{V}} d_{pq} x_{pq} \tag{2}$$

$$\text{s.t.} \sum_{q \in \mathcal{V}} x_{pq} = 1 \tag{3}$$

$$x_{pq} \leq x_{qq} \tag{4}$$

$$x_{pq} \geq 0 \tag{5}$$

To get an equivalent problem to (1), we simply have to replace $x_{pq} \geq 0$ with $x_{pq} \in \{0,1\}$. In this case, each binary variable $x_{pq}$ with $p \neq q$ indicates whether object $p$ has been assigned to cluster center $q$ or not, while binary variable $x_{qq}$ indicates whether object $q$ has been chosen as a cluster center or not. Constraints (3) simply express the fact that each object must be assigned to exactly one center, while constraints (4) require that if $p$ has been assigned to $q$ then object $q$ must obviously be chosen as a center.

Obviously at the core of any clustering problem of this type lies the issue of deciding which objects will be chosen as centers. To deal with that, a key idea of our approach is to rely on, what we call, the *stability* of an object. This will be a well defined measure which, intuitively, tries to quantitatively answer the following question: *"How much do we need to penalize an object in order to ensure that it is never selected as an optimal cluster center?"* For formalizing this concept, we will make use of the LP relaxation PRIMAL. We will thus define the stability $S(q)$ of an object $q$ as follows:

$$S(q) = \inf\{\text{perturbation } s \text{ that has to be applied to penalty } d_{qq} \text{ (i.e., } d_{qq} \leftarrow d_{qq} + s) \quad (6)$$
$$\text{such that PRIMAL has no optimal solution } \mathbf{x} \text{ with } x_{qq} > 0\}$$

An object $q$ can be stable or unstable depending on whether it holds $S(q) \geq 0$ or $S(q) < 0$. To select a set of centers $\mathcal{Q}$, we will then rely on the following observation: *a stable object with high stability is also expected to be, with high probability, an optimal center in* (1). The reason is that the assumption of a high $S(q) \geq 0$ is essentially a very strong requirement (much stronger than simply requiring $q$ to be active in the relaxed problem PRIMAL): it further requires that $q$ will be active for all problems PRIMAL$(d_{qq}+s)$[1] as well (where $s \leq S(q)$). Hence, our strategy for generating $\mathcal{Q}$ will be to sequentially select a set of stable objects, trying, at each step, to select an object of approximately maximum stability (as already explained, there is high chance that this object will be an optimal center in (1)). Furthermore, each time we insert a stable object $q$ to $\mathcal{Q}$, we reestimate stabilities for the remaining objects in order to take this fact into account (e.g., an object may become unstable if we know that it holds $x_{qq} = 1$ for another object $q$). To achieve that, we will need to impose extra constraints to PRIMAL (as we shall see, this will help us to obtain an accurate estimation for the stabilities of the remaining objects given that objects in $\mathcal{Q}$ are already chosen as centers). Of course, this process repeats until no more stable objects can be found.

## 2.1 Margins and dual-based clustering

For having a practical algorithm, the most critical issue is how to obtain a rough approximation to the stability of an object $q$ in a computationally efficient manner. As we shall see, to achieve this we will need to to move to the dual domain and introduce a novel concept that lies at the core of our approach: the *margin* of dual solutions. But, first, we need to introduce the dual to problem PRIMAL, which is the linear program called DUAL in (7)[2]:

$$\text{DUAL} \equiv \max\ D(\mathbf{h}) = \sum_{p \in \mathcal{V}} h_p \quad (7)$$

$$\text{s.t. } h_p = \min_{q \in \mathcal{V}} h_{pq}, \qquad \forall p \in \mathcal{V} \quad (8)$$

$$\sum_{p \in \mathcal{V}} h_{pq} = \sum_{p \in \mathcal{V}} d_{pq}, \quad \forall q \in \mathcal{V} \quad (9)$$

$$h_{pq} \geq d_{pq} \qquad \forall p \neq q \quad (10)$$

Dual variables $h_{pq}$ can be thought of as representing *pseudo-distances* between objects, while each variable $h_p$ represents the minimum pseudo-distance from $p$ (which is, in fact, 'thought' by the dual as an estimation of the actual distance between $p$ and its closest active center).

Given a feasible dual solution $\mathbf{h}$, we can now define its *margin* $\Delta_q(\mathbf{h})$ (with respect to object $q$) as follows:

$$\Delta_q(\mathbf{h}) = \sum_{p:h_{pq}=h_p} (\hat{h}_p - h_p) - \sum_{p \neq q} (h_{pq} - \max(h_p, d_{pq})) - \left(h_{qq} - h_q\right), \quad (11)$$

where (for any $\mathbf{h}$) $\hat{h}_p$ hereafter denotes the next-to-minimum pseudo-distance from $p$.

There is a very tight connection between margins of dual solutions and stabilities of objects. The following lemma provides a first indication for this fact and shows that we can actually use margins to decide whether an object is stable or not and also to lower bound or upper bound its stability accordingly (see [7] for proofs):

**Lemma 1** ([7]). *Let $\mathbf{h}$ be an optimal dual solution to* DUAL.

1. *If* $\Delta_q(\mathbf{h}) > 0$ *then* $S(q) \geq \Delta_q(\mathbf{h})$.

2. *If* $\Delta_q(\mathbf{h}) < 0$ *then* $S(q) \leq \Delta_q(\mathbf{h})$.

In fact, the following fundamental theorem goes even further by proving that stabilities can be fully characterized solely in terms of margins. Hence, margins and stabilities are two concepts that can be roughly considered as dual to each other:

**Theorem 2** ([7]). *The following equalities hold true:*

$$S(q) \geq 0 \Rightarrow S(q) = \sup\{\Delta_q(\mathbf{h}) \mid \mathbf{h} \text{ optimal solution to } \text{DUAL}\} \,, \tag{12}$$

$$S(q) \leq 0 \Rightarrow S(q) = \inf\{\Delta_q(\mathbf{h}) \mid \mathbf{h} \text{ optimal solution to } \text{DUAL}\} \,. \tag{13}$$

*Furthermore, it can be shown that:*

$$S(q) = sign(S(q)) \cdot \sup\{|\Delta_q(\mathbf{h})| \ \Big| \ \mathbf{h} \text{ optimal solution to } \text{DUAL}\} \,. \tag{14}$$

What the above theorem essentially tells us is that one can compute $S(q)$ exactly, simply by considering the margins of optimal dual solutions. Based on this fact, it is therefore safe to assume that solutions $\mathbf{h}$ with high (but not necessarily maximum) dual objective $D(\mathbf{h})$ will have margins that are good approximations to $S(q)$, i.e., it holds:

$$S(q) \approx \Delta_q(\mathbf{h}) \,. \tag{15}$$

This is exactly the idea that our clustering algorithm will rely on in order to efficiently discover objects that are stable. It thus maintains a dual solution $\mathbf{h}$ and a set $\mathcal{Q}$ containing all stable objects chosen as centers up to the current point ($\mathcal{Q}$ is empty initially). At each iteration, it increases the dual objective $D(\mathbf{h})$ by updating solution $\mathbf{h}$ via an operation called DISTRIBUTE. This operation is repeatedly applied until a high enough objective value $D(\mathbf{h})$ is obtained such that at least one stable object is revealed based on the estimated margins of $\mathbf{h}$. At that point, the set $\mathcal{Q}$ is expanded and $\mathbf{h}$ is updated (via an operation called PROJECT) to take account of this fact. The process is then repeated until no more stable objects can be found. A remarkable thing to note in this process is that, as we shall see, determining how to update $\mathbf{h}$ during the DISTRIBUTE operation (i.e., for increasing the dual objective) also relies critically on the use of margins.

Another technical point that we need to solve comes from the fact that $\mathcal{Q}$ gets populated with objects as the algorithm proceeds, which is something that we certainly need to take into account when estimating object stabilities. Fortunately, there is a very elegant solution to this problem: since all objects in $\mathcal{Q}$ are assumed to be cluster centers (i.e., it holds $x_{qq} = 1$, $\forall q \in \mathcal{Q}$), instead of working with problems PRIMAL and DUAL, it suffices that one works with the following primal-dual pair of LPs called $\text{PRIMAL}_\mathcal{Q}$ and $\text{DUAL}_\mathcal{Q}$[3]:

$$
\begin{array}{ll}
\text{PRIMAL}_\mathcal{Q} = \min \ \text{PRIMAL} & \text{DUAL}_\mathcal{Q} = \max \ \text{DUAL} \\
\qquad\quad \text{s.t. } x_{qq} = 1, \ \forall q \in \mathcal{Q} & \qquad\quad \text{s.t. } h_{pq} = d_{pq}, \ \forall \{p, q\} \cap \mathcal{Q} \neq \emptyset
\end{array}
$$

This means, e.g., that stability $S(q)$ is now defined by using $\text{PRIMAL}_\mathcal{Q}$ (instead of PRIMAL) in (6). Likewise, lemma 1 and theorem 2 still continue to hold true provided that DUAL is replaced with $\text{DUAL}_\mathcal{Q}$ in the statement of these theorems. In addition to that, the definition of margin $\Delta_q(\mathbf{h})$ needs to be modified as follows :

$$\Delta_q(\mathbf{h}) = \sum_{p \notin \mathcal{Q}: h_{pq} = h_p} (\hat{h}_p - h_p) - \sum_{p \notin \mathcal{Q} \cup \{q\}} (h_{pq} - \max(h_p, d_{pq})) - \left(h_{qq} - h_q\right). \tag{16}$$

**The PROJECT operation:** Given this modified definition of margins, we can now update $\mathcal{Q}$ at any iteration in the following manner:

$$\text{EXPAND: Compute } \bar{q} = \arg\max_{q \notin Q} \Delta_q(\mathbf{h}) \text{ and if } \Delta_{\bar{q}}(\mathbf{h}) \geq 0 \text{ then set } \mathcal{Q} = \mathcal{Q} \cup \{\bar{q}\} \,. \tag{17}$$

Based on the fact that margins are used as approximations to the stabilities of objects, the above update simply says that the object $\bar{q}$ with maximum stability should be chosen as the new center at the current iteration, provided of course that this object $\bar{q}$ is stable. Furthermore, in this case, we also

```
1: h ← d;
2: while max_{q∉Q} Δ_q(h) < 0 do
3:     D_prev ← D(h);   h ← DISTRIBUTE(h);
4:     if D_prev = D(h) then exit;
5: end
6: q̄ ← arg max_{q∉Q} Δ_q(h);   Q ← Q ∪ {q̄};   h ← PROJECT(h);
7: goto 2;
```

**Fig. 1:** Pseudocode of our clustering algorithm.

need to update the current dual solution $\mathbf{h}$ in order to take account of the fact that extra constraints have been added to $\text{DUAL}_Q$ (these are a result of the extra constraint $x_{\bar{q}\bar{q}} = 1$ that has been added to $\text{PRIMAL}_Q$). By definition of $\text{DUAL}_Q$, the new constraints are $h_{\bar{q}p} = d_{\bar{q}p}$, $h_{p\bar{q}} = d_{p\bar{q}}$ for all $p \notin Q$ and, so, one has to apply the following operation, which simply projects the current dual solution into the feasible set of the updated linear program $\text{DUAL}_Q$:

$$\text{PROJECT:} \quad h_{pp} \mathrel{+}= h_{\bar{q}p} - d_{\bar{q}p}, \quad h_{\bar{q}p} = d_{\bar{q}p}, \quad h_{p\bar{q}} = d_{p\bar{q}}, \quad \forall p \notin Q . \tag{18}$$

Note that update $h_{pp} \mathrel{+}= h_{\bar{q}p} - d_{\bar{q}p}$ is needed for maintaining dual feasibility constraint (9). Essentially, PROJECT is a warm-start operation, that allows us to reuse existing information for computing a solution $\mathbf{h}$ that has a high dual objective value $D(\mathbf{h})$ and is also feasible to the updated $\text{DUAL}_Q$.

**The DISTRIBUTE operation:** In case it holds $\Delta_q(\mathbf{h}) < 0$ for all $q \notin Q$, this means that we are unable to find an object with good stability at the current iteration. To counter that, we will thus need to update solution $\mathbf{h}$ in order to increase its dual objective value (recall that, by lemma 1, stable objects will necessarily be revealed at an optimal dual solution, i.e., at a dual solution of maximum objective). Intuitively, what happens is that as we increase the dual objective $D(\mathbf{h})$, objects not in $Q$ actually try to compete with each other for achieving a large margin. Interestingly enough, in order to increase $D(\mathbf{h})$, we will again have to rely on the margins of the current dual solution. In particular, it turns out that, if $\Delta_q(\mathbf{h}) < 0$ holds true for all $q \notin Q$, then the following very simple update of $\mathbf{h}$ is guaranteed to increase the dual objective:

$$\text{DISTRIBUTE:} \ \forall p, q \notin Q, \ h_{pq} = \begin{cases} \max(h_p, d_{pq}), & \text{if } p \neq q \text{ AND } \left( p \in \mathcal{L}_Q \text{ OR } h_p < d_{pq} \right) \\ h_p - \frac{\Delta_q(\mathbf{h})}{|\mathcal{V}_q|}, & \text{else if } h_{pq} > h_p \\ \hat{h}_p - \frac{\Delta_q(\mathbf{h})}{|\mathcal{V}_q|}, & \text{else if } h_{pq} = h_p \end{cases}$$

In the above update, we denote by $\mathcal{L}_Q$ the set of objects whose minimum pseudo-distance $h_p$ is attained at an object from $Q$, i.e., $\mathcal{L}_Q = \{p \notin Q \,|\, h_p = \min_{q \in Q} h_{pq}\}$, while $|\mathcal{V}_q|$ denotes the cardinality of the set $\mathcal{V}_q = \{p \notin Q \cup \mathcal{L}_Q \,|\, h_p \geq d_{pq}\} \cup \{q\}$. The following theorem then holds true:

**Theorem 3.** *If $\max_{q\notin Q} \Delta_q(\mathbf{h}) < 0$, then the DISTRIBUTE operation maintains feasibility and, unless $\mathcal{V} = Q \cup \mathcal{L}_Q$, it also strictly increases the dual objective.*

The pseudocode of the resulting algorithm is shown in Fig. 1. As already explained, it is an iterative algorithm, which keeps updating a dual solution $\mathbf{h}$ by using the DISTRIBUTE and PROJECT operations (the latter applied only when needed) until the dual objective can no longer increase. Note also that, besides maintaining a dual solution $\mathbf{h}$, the algorithm also maintains $Q$ which provides a current clustering and also has a primal cost $E(Q)$. With respect to this cost, the following theorem can be shown to hold true:

**Theorem 4.** *If $\max_{q\notin Q} \Delta_q(\mathbf{h}) > 0$, then the EXPAND operation strictly decreases the primal cost $E(Q)$.*

This implies that the sequence of primal costs $E(Q)$ generated by the algorithm is decreasing (recall that we actually want to minimize $E(\cdot)$). It is worth noting at this point that nowhere have we tried to enforce this property by explicitly considering the primal cost when updating $Q$. This is achieved simply thanks to the requirement of always selecting objects with high stability, thus showing how powerful this requirement actually is. We also note that the algorithm's convergence is always guaranteed: the algorithm terminates when neither the primal cost $E(Q)$ decreases nor the dual objective $D(\mathbf{h})$ increases during the current iteration. Finally, we note that exactly the same algorithm applies to the general case where the objects in $\mathcal{V}$ form a graph with edges $\mathcal{E}$ (distance $d_{pq}$ is then defined only for $pq \in \mathcal{E}$). In this case, it is easy to verify that the cost of each iteration will be $O(|\mathcal{E}|)$. Furthermore, the algorithm converges extremely fast in practice (i.e. in very few iterations).

## 3   Related work

Before proceeding, let us briefly mention how our method relates to some state-of-the-art exemplar-based clustering techniques. Affinity propagation [5] is a recently proposed method for clustering, which relies on minimizing exactly the same objective function (1). This is an iterative algorithm, which repeatedly updates (through messages) the so-called responsibilities and availabilities. These can be considered as counterparts to our pseudo-distances $h_{pq}$. Affinity propagation also estimates the so-called self-availabilities for measuring the likelihood of an object being a cluster center. On the contrary, we use for the same purpose the margins that approximate the stability of an object. Furthermore, compared to affinity propagation, our method offers the following significant advantages: its convergence is always guaranteed, it is parameter-free (no need for adjusting parameters such as damping factors in order to ensure convergence), it is a descent method (objective function (1) always decreases), and it can make use of the computed dual solutions for deriving online optimality bounds for free (these can be used for assessing that the derived solutions are almost optimal). At the same time, our method performs equally well or better in practice. Very recently, another exemplar-based algorithm has been proposed as well, which relies on solving a convex formulation of clustering [8]. We note, however, that this method is used for solving a different and much easier problem, which is that of soft clustering. Furthermore, it relies on a convex relaxation which is known to be much less tight than the LP relaxation PRIMAL we use here (essentially [8] replaces all constraints $x_{pq} \le x_{qq}, \forall p \in \mathcal{V}$ with the much looser constraint $\sum_p x_{pq} \le |\mathcal{V}| \cdot x_{qq}$). As a result, generated solutions are expected to be of much lower quality. We also note that, unlike EM-like clustering algorithms such as K-means, our method is totally insensitive to initialization conditions and does not get stuck at bad local minima (thus yielding solutions of much better quality). Also, it is much more efficient than methods like [6], that require solving very large linear programs.

## 4   Experimental results

To illustrate the robustness of our algorithm to noise and its insensitivity to initialization, we start by showing clustering results on synthetic data. The synthetic datasets were generated using the following procedure: 2D points were sampled from a mixture of gaussian distributions, where the centers of the gaussians were arranged in an approximately grid-like fashion over the plane. In addition to that, random outliers were generated uniformly all over the grid, with their number being equal to half the number of the points drawn from the gaussian distributions. One such dataset (consisting of 24 gaussians) is displayed in Fig. 2, where colored crosses correspond to samples from gaussians, while the black dots correspond to outliers. The clustering result produced by our algorithm is shown in Fig. 2(a). As can be seen from that figure, despite the heavy percentage of noise, our method has been able to accurately detect all gaussian centers and successfully cluster this 2D dataset. Note that the number of gaussians was not given as input to our algorithm. Instead, it was inferred based on a common penalty term $d_{qq}$ for all objects $q$, which was set roughly equal to the median distance between points. On the contrary, K-means was unable to produce a good result for this dataset despite the fact that it was restarted multiple times (100 runs were used in this case). This is, of course, due to its well known sensitivity to initialization conditions. We repeated multiple experiments by varying the number of gaussians. Contrary to our algorithm, behavior of K-means gets even worse as this number increases.

We have also plotted in Fig. 2(c) the primal and dual costs that were generated by our algorithm when it was applied to the example of Fig. 2(a). These correspond to the solid red and dashed blue curves respectively. Note that the dual costs represent lower bounds to the optimum value of the objective function $E(\cdot)$, while the primal costs represent obviously upper bounds. This fact allows us to obtain online optimality bounds with respect to how far our current primal solution $\mathcal{Q}$ is with respect to the unknown optimum of $E(\cdot)$. These bounds are, of course, refined continuously as the algorithm proceeds and can be useful for assessing its performance. For instance, in this particular example, we can be sure that the primal cost of our final solution is within 1% of the unknown optimum of function $E(\cdot)$, i.e., an approximately optimal solution has been obtained.

Next we show some results from applying our algorithm to the challenging problem of multibody 3D segmentation, which has several applications in computer vision. As we shall see, a non-Euclidean distance for clustering will have to be used in this case. According to the 3D segmentation problem, we are given a set of $N$ pixel correspondences between two images. These correspondences result

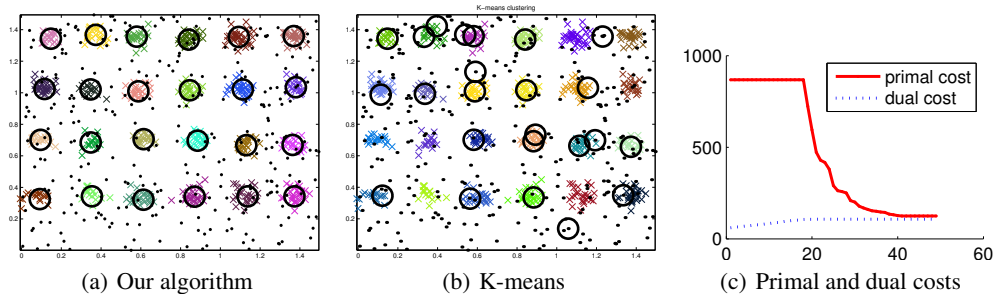

| (a) Our algorithm | (b) K-means | (c) Primal and dual costs |

**Fig. 2:** Clustering results for synthetic data. The centers of the big circles represent the points chosen as cluster centers by the 2 algorithms. The primal and dual costs in (c) verify that the cost of our algorithm's solution is within 1% of the optimum cost.

from $K$ objects undergoing $K$ 3D rigid-body motions relative to a moving camera. The 3D-motion segmentation problem is the task of clustering these $N$ pixel pairs according to the $K$ moving objects. We consider the more general and difficult scenario of a fully projective camera model. In this case, each pixel pair, say, $p_i = (y_i, z_i)$ that belongs to a moving object $k$ should satisfy an epipolar constraint:

$$y_i^T F_k z_i = 0 \ , \tag{19}$$

where $F_k$ represents the fundamental matrix associated with the k-th 3D motion. Of course, the matrices $F_k$ corresponding to different motions are unknown to us. Hence, to solve the 3D segmentation problem, we need to estimate both the matrices $F_k$ as well as the association of each pixel pair $p_i = (y_i, z_i)$ to the correct fundamental matric $F_k$. To this end, we sample a large set of fundamental matrices by using a RANSAC-based scheme (we recall that a random set of, e.g., 8 pixel pairs $p_i$ is enough for generating a new fundamental matrix). The resulting matrices, say, $\{F_k\}$ will then correspond to cluster centers, whereas all the input pixel pairs $\{p_i\}$ will correspond to objects that need to be assigned to an active cluster center. A clustering objective function of the form (1) thus results and by minimizing it we can also obtain a solution to the 3D segmentation problem. Of course, in this case, the distance function $d(p_i, F_k)$ between an object $p_i = (y_i, z_i)$ and a cluster center will not be Euclidean. Instead, based on (19), we can use a distance of the following form:

$$d(p_i, F_k) = |y_i^T F_k z_i| \ . \tag{20}$$

Due to being more robust, a normalized version of the above distance is usually preferred in practice. Figure 3 displays 3D motion segmentation results that were obtained by applying our algorithm to two image pairs (points with different colors correspond to different motions). These examples were downloaded from a publicly available motion segmentation database [9] with ground-truth. The ground-truth motion segmentation is also shown for each example and, as can be seen, it is almost identical with the segmentation estimated by our algorithm.

We next compare our method to Affinity Propagation (AP). Some really impressive results on 4 very challenging datasets have been reported for that algorithm in [5], indicating that it outperforms any other center-based clustering method. In particular, AP has been used for: clustering images of faces (using the squared error distance), detecting genes in microarray data (using a distance based on exons' transcriptions levels), identifying representative sentences in manuscripts (using

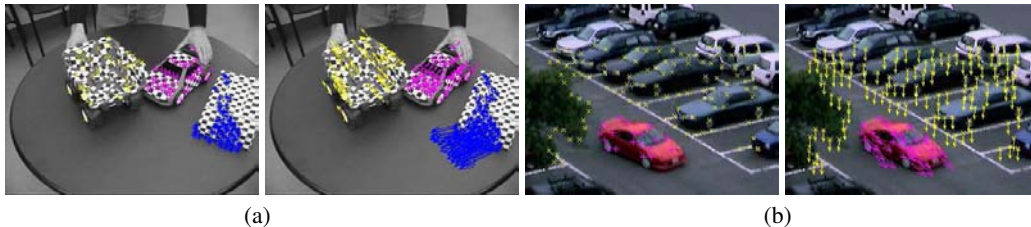

| (a) | (b) |

**Fig. 3:** Two 3D motion segmentation results. For each one we show (left) ground truth segmentation of feature points and (right) estimated segmentation along with the input optical flow vectors.

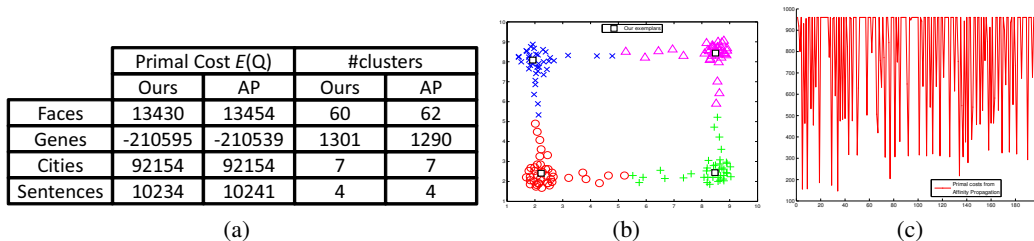

| | Primal Cost $E$(Q) | | #clusters | |
|---|---|---|---|---|
| | Ours | AP | Ours | AP |
| Faces | 13430 | 13454 | 60 | 62 |
| Genes | -210595 | -210539 | 1301 | 1290 |
| Cities | 92154 | 92154 | 7 | 7 |
| Sentences | 10234 | 10241 | 4 | 4 |
| (a) | | (b) | | (c) |

**Fig. 4: (a)** Comparison of our algorithm with affinity propagation [5] on the 4 very challenging datasets 'Faces', 'Genes', 'Cities' and 'Sentences' from [5]. Since the goal of both algorithms is to minimize objective function $E(\mathcal{Q})$, for each dataset we report the final value of this function and the number of estimated clusters. We have used exactly the same settings for both methods. **(b)** Our algorithm's clustering when applied to the 'four-clouds' dataset from [1]. The primal costs generated by AP for this dataset (shown in **(c)**) demonstrate that AP fails to converge in this case (to prevent that, a properly chosen damping factor has to be used).

the relative entropy as distance), and identifying cities that can be easily accessed by airline travel. In Fig. 4(a), we compare our method to AP on these publicly available problems. Since both methods rely on optimizing the same objective function, we list the values obtained by the two methods for the corresponding problems. Exactly the same settings have been used for both algorithms, with AP using the parameters proposed in [5]. Note that in all cases our algorithm manages to obtain a solution of equal or lower value than AP. This is true even, e.g., in the Genes dataset, where a higher number of clusters is selected by our algorithm (and thus a higher penalty for activating them is paid). Furthermore, an additional advantage of our algorithm is that, unlike AP, it is always guaranteed to converge (e.g., see Figs 4(b), 4(c)). We note that, due to lack of space, a running time comparison with AP, as well as a comparison of our algorithm to the method in [10], are included in [7].

## 5   Conclusions

In this paper we have introduced a very powerful and efficient center-based clustering algorithm, derived from LP duality theory. The resulting algorithm has guaranteed convergence and can handle data sets with arbitrary distance functions. Furthermore, despite its extreme generality, the proposed method is insensitive to initialization and computes clusterings of very low cost. As such, and considering the key role that clustering has in many problems, we believe that our method can find use in a wide variety of tasks. As another very important (both practical and theoretical) contribution of this work we also consider the fact of introducing the notions of LP-based stabilities and margins, two quantities that, as we have proved, are dual to each other and can be used for deciding what objects should be chosen as cluster centers. We strongly believe that these ideas can be of both practical and theoretical interest not just for designing center-based clustering algorithms, but also in many other contexts as well.

## Footnotes

[1]PRIMAL$(z)$ denotes a modified problem PRIMAL where the penalty for $q$ has been set equal to $z$.

[2]Problem DUAL results from the standard dual to PRIMAL after applying a transformation to the dual variables.

[3]Actually, to represent the dual of $\text{PRIMAL}_\mathcal{Q}$ exactly, we need to add a constant in the objective function of $\text{DUAL}_\mathcal{Q}$. Since, however, this constant does not affect maximization, it is thus omitted for clarity.

## References

[1] A. Ng, M. Jordan, and Y. Weiss, "On spectral clustering: Analysis and an algorithm," in *NIPS*, 2001.

[2] D. Verma and M. Meila, "A comparison of spectral clustering algorithms," Tech. Rep., 2001.

[3] A. Banerjee, S. Merugu, I. S. Dhillon, and J. Ghosh, "Clustering with bregman divergences," *J. Mach. Learn. Res.*, vol. 6, pp. 1705–1749, 2005.

[4] B. Fischer, V. Roth, and J. Buhmann, "Clustering with the connectivity kernel," in *NIPS*, 2004.

[5] B. J. Frey and D. Dueck, "Clustering by passing messages between data points," *Science*, vol. 315, 2007.

[6] M. Charikar, S. Guha, É. Tardos, and D. B. Shmoys, "A constant-factor approximation algorithm for the k-median problem," *J. Comput. Syst. Sci.*, vol. 65, no. 1, pp. 129–149, 2002.

[7] N. Komodakis, N. Paragios, and G. Tziritas, "Clustering via LP-based Stabilities," Tech. Report, 2009.

[8] D. Lashkari and P. Golland, "Convex clustering with exemplar-based models," in *NIPS*, 2008.

[9] R. Tron and R. Vidal, "A benchmark for the comparison of 3-d motion segmentation algorithms," in *CVPR*, 2007.

[10] M. Leone, Sumedha, and M. Weigt, "Clustering by soft-constraint affinity propagation: applications to gene-expression data," *Bioinformatics*, vol. 23, no. 20, pp. 2708–2715, 2007.

